# Multi-Stage Dantzig Selector

**Ji Liu, Peter Wonka, Jieping Ye**
Arizona State University
{ji.liu,peter.wonka,jieping.ye}@asu.edu

## Abstract

We consider the following sparse signal recovery (or feature selection) problem: given a design matrix $X \in \mathbb{R}^{n \times m}$ ($m \gg n$) and a noisy observation vector $y \in \mathbb{R}^n$ satisfying $y = X\beta^* + \epsilon$ where $\epsilon$ is the noise vector following a Gaussian distribution $N(0, \sigma^2 I)$, how to recover the signal (or parameter vector) $\beta^*$ when the signal is sparse?

The Dantzig selector has been proposed for sparse signal recovery with strong theoretical guarantees. In this paper, we propose a multi-stage Dantzig selector method, which iteratively refines the target signal $\beta^*$. We show that if $X$ obeys a certain condition, then with a large probability the difference between the solution $\hat{\beta}$ estimated by the proposed method and the true solution $\beta^*$ measured in terms of the $l_p$ norm ($p \geq 1$) is bounded as

$$\|\hat{\beta} - \beta^*\|_p \leq \left( C(s-N)^{1/p}\sqrt{\log m} + \Delta \right)\sigma,$$

where $C$ is a constant, $s$ is the number of nonzero entries in $\beta^*$, $\Delta$ is independent of $m$ and is much smaller than the first term, and $N$ is the number of entries of $\beta^*$ larger than a certain value in the order of $\mathcal{O}(\sigma\sqrt{\log m})$. The proposed method improves the estimation bound of the standard Dantzig selector approximately from $Cs^{1/p}\sqrt{\log m}\sigma$ to $C(s-N)^{1/p}\sqrt{\log m}\sigma$ where the value $N$ depends on the number of large entries in $\beta^*$. When $N = s$, the proposed algorithm achieves the oracle solution with a high probability. In addition, with a large probability, the proposed method can select the same number of correct features under a milder condition than the Dantzig selector.

## 1 Introduction

The sparse signal recovery problem has been studied in many areas including machine learning [18, 19, 22], signal processing [8, 14, 17], and mathematics/statistics [2, 5, 7, 10, 11, 12, 13, 20]. In the sparse signal recovery problem, one is mainly interested in the signal recovery accuracy, i.e., the distance between the estimation $\hat{\beta}$ and the original signal or the true solution $\beta^*$. If the design matrix $X$ is considered as a feature matrix, i.e., each column is a feature vector, and the observation $y$ as a target object vector, then the sparse signal recovery problem is equivalent to feature selection (or model selection). In feature selection, one concerns the feature selection accuracy. Typically, a group of features corresponding to the coefficient values in $\hat{\beta}$ larger than a threshold form the supporting feature set. The difference between this set and the true supporting set (i.e., the set of features corresponding to nonzero coefficients in the original signal) measures the feature selection accuracy.

Two well-known algorithms for learning sparse signals include LASSO [15] and Dantzig selector [7]:

$$\textbf{LASSO} \quad \min_{\beta} : \frac{1}{2}\|X\beta - y\|_2^2 + \lambda\|\beta\|_1 \tag{1}$$

$$\text{Dantzig Selector} \quad \min_{\beta} : ||\beta||_1$$

$$s.t. : ||X^T(X\beta - y)||_\infty \leq \lambda \tag{2}$$

Strong theoretical results concerning LASSO and Dantzig selector have been established in the literature [4, 5, 7, 17, 20, 22].

## 1.1 Contributions

In this paper, we propose a multi-stage procedure based on the Dantzig selector, which estimates the supporting feature set $F_0$ and the signal $\hat{\beta}$ iteratively. The intuition behind the proposed multi-stage method is that feature selection and signal recovery are tightly correlated and they can benefit from each other: a more accurate estimation of the supporting features can lead to a better signal recovery and a more accurate signal recovery can help identify a better set of supporting features. In the proposed method, the supporting set $F_0$ starts from an empty set and its size increases by one after each iteration. At each iteration, we employ the basic framework of Dantzig selector and the information about the current supporting feature set $F_0$ to estimate the new signal $\hat{\beta}$. In addition, we select the supporting feature candidates in $F_0$ among all features in the data at each iteration, thus allowing to remove incorrect features from the previous supporting feature set.

The main contributions of this paper lie in the theoretical analysis of the proposed method. Specifically, we show: 1) the proposed method can improve the estimation bound of the standard Dantzig selector approximately from $Cs^{1/p}\sqrt{\log m}\sigma$ to $C(s-N)^{1/p}\sqrt{\log m}\sigma$ where the value $N$ depends on the number of large entries in $\beta^*$; 2) when $N = s$, the proposed algorithm can achieve the oracle solution with a high probability; 3) with a high probability, the proposed method can select the same number of correct features under a milder condition than the standard Dantzig selector method. The numerical experiments validate these theoretical results.

## 1.2 Related Work

Sparse signal recovery without the observation noise was studied in [6]. It has been shown that under certain irrepresentable conditions, the 0-support of the LASSO solution is consistent with the true solution. It was shown that when the absolute value of each element in the true solution is large enough, a weaker condition (coherence property) can guarantee the feature selection accuracy [5]. The prediction bound of LASSO, i.e., $||X(\hat{\beta} - \beta^*)||_2$, was also presented. A comprehensive analysis for LASSO, including the recovery accuracy in an arbitrary $l_p$ norm ($p \geq 1$), was presented in [20]. In [7], the Dantzig selector was proposed for sparse signal recovery and a bound of recovery accuracy with the same order as LASSO was presented. An approximate equivalence between the LASSO estimator and the Dantzig selector was shown in [1]. In [11], the $l_\infty$ convergence rate was studied simultaneously for LASSO and Dantzig estimators in a high-dimensional linear regression model under a mutual coherence assumption. In [9], conditions on the design matrix $X$ under which the LASSO and Dantzig selector coefficient estimates are identical for certain tuning parameters were provided.

Many heuristic methods have been proposed in the past, including greedy least squares regression [16, 8, 19, 21, 3], two stage LASSO [20], multiple thresholding procedures [23], and adaptive LASSO [24]. They have been shown to outperform the standard convex methods in many practical applications. It was shown [16] that under an irrepresentable condition the solution of the greedy least squares regression algorithm (also named OMP or forward greedy algorithm) guarantees the feature selection consistency in the noiseless case. The results in [16] were extended to the noisy case [19]. Very recently, the results were further improved in [21] by considering arbitrary loss functions (not necessarily quadratic). In [3], the consistency of OMP was shown under the mutual incoherence conditions. A multiple thresholding procedure was proposed to refine the solution of LASSO or Dantzig selector [23]. An adaptive forward-backward greedy algorithm was proposed [18], and it was shown that under the restricted isometry condition the feature selection consistency is achieved if the minimal nonzero entry in the true solution is larger than $\mathcal{O}(\sigma\sqrt{\log m})$. The adaptive LASSO was proposed to adaptively tune the weight value for the $L_1$ penalty, and it was shown to enjoy the oracle properties [24].

## 1.3 Definitions, Notations, and Basic Assumptions

We use $X \in \mathbb{R}^{n \times m}$ to denote the design matrix and focus on the case $m \gg n$, i.e., the signal dimension is much larger than the observation dimension. The correlation matrix $A$ is defined as $A = X^T X$ with respect to the design matrix. The noise vector $\epsilon$ follows the multivariate normal distribution $\epsilon \sim N(0, \sigma^2 I)$. The observation vector $y \in \mathbb{R}^n$ satisfies $y = X\beta^* + \epsilon$, where $\beta^*$ denotes the original signal (or true solution). $\hat{\beta}$ is used to denote the solution of the proposed algorithm. The $\alpha$-supporting set ($\alpha \geq 0$) for a vector $\beta$ is defined as

$$supp_\alpha(\beta) = \{j : |\beta_j| > \alpha\}.$$

The "supporting" set of a vector refers to the 0-supporting set. $F$ denotes the supporting set of the original signal $\beta^*$. For any index set $S$, $|S|$ denotes the size of the set and $\bar{S}$ denotes the complement of $S$ in $\{1, 2, 3, ..., m\}$. In this paper, $s$ is used to denote the size of the supporting set $F$, i.e., $s = |F|$. We use $\beta_S$ to denote the subvector of $\beta$ consisting of the entries of $\beta$ in the index set $S$. The $l_p$ norm of a vector $v$ is computed by $\|v\|_p = (\sum_i v_i^p)^{1/p}$, where $v_i$ denotes the $i$th entry of $v$. The oracle solution $\bar{\beta}$ is defined as $\bar{\beta}_F = (X_F^T X_F)^{-1} X_F^T y$, and $\bar{\beta}_{\bar{F}} = 0$. We employ the following notation to measure some properties of a PSD matrix $M \in \mathbb{R}^{K \times K}$ [20]:

$$\mu_{M,k}^{(p)} = \inf_{u \in \mathbb{R}^k, |I|=k} \frac{\|M_{I,I}u\|_p}{\|u\|_p}, \quad \rho_{M,k}^{(p)} = \sup_{u \in \mathbb{R}^k, |I|=k} \frac{\|M_{I,I}u\|_p}{\|u\|_p}, \tag{3}$$

$$\theta_{M,k,l}^{(p)} = \sup_{u \in \mathbb{R}^l, |I|=k, |J|=l, I \cap J = \varnothing} \frac{\|M_{I,J}u\|_p}{\|u\|_p}, \tag{4}$$

where $p \in [1, \infty]$, $I$ and $J$ are disjoint subsets of $\{1, 2, ..., K\}$, and $M_{I,J} \in \mathbb{R}^{|I| \times |J|}$ is a submatrix of $M$ with rows from the index set $I$ and columns from the index set $J$. Additionally, we use the following notation to denote two probabilities:

$$\eta_1' = \eta_1 (\pi \log((m-s)/\eta_1))^{-1/2}, \quad \eta_2' = \eta_2 (\pi \log(s/\eta_2))^{-1/2}. \tag{5}$$

where $\eta_1$ and $\eta_2$ are two factors between 0 and 1. In this paper, if we say "large", "larger" or "the largest", it means that the absolute value is large, larger or the largest. For simpler notation in the computation of sets, we sometimes use "$S_1 + S_2$" to indicate the union of two sets $S_1$ and $S_2$, and use "$S_1 - S_2$" to indicate the removal of the intersection of $S_1$ and $S_2$ from the first set $S_1$. In this paper, the following assumption is always admitted.

**Assumption 1.** *We assume that $s = |supp_0(\beta^*)| < n$, the variable number is much larger than the feature dimension (i.e. $m \gg n$), each column vector is normalized as $X_i^T X_i = 1$ where $X_i$ indicates the $i$th column (or feature) of $X$, and the noise vector $\epsilon$ follows the Gaussian distribution $N(0, \sigma^2 I)$.*

In the literature, it is often assumed that $X_i^T X_i = n$, which is essentially identical to our assumption. However, this may lead to a slight difference of a factor $\sqrt{n}$ in some conclusions. We have automatically transformed conclusions from related work according to our assumption when citing them in our paper.

## 1.4 Organization

The rest of the paper is organized as follows. We present our multi-stage algorithm in Section 2. The main theoretical results are summarized in Section 3 with detailed proofs given in the supplemental material. The numerical simulation is reported in Section 4. Finally, we conclude the paper in Section 5. All proofs can be found in the supplementary file.

## 2 The Multi-Stage Dantzig Selector Algorithm

In this section, we introduce the multi-stage Dantzig selector algorithm. In the proposed method, we update the support set $F_0$ and the estimation $\hat{\beta}$ iteratively; the supporting set $F_0$ starts from an empty set and its size increases by one after each iteration. At each iteration, we employ the basic

framework of Dantzig selector and the information about the current supporting set $F_0$ to estimate the new signal $\hat{\beta}$ by solving the following linear program:

$$
\begin{aligned}
\min \ & \|\beta_{\bar{F}_0}\|_1 \\
s.t. \ & \|X_{\bar{F}_0}^T(X\beta - y)\|_\infty \le \lambda \\
& \|X_{F_0}^T(X\beta - y)\|_\infty = 0.
\end{aligned}
\tag{6}
$$

Since the features in $F_0$ are considered as the supporting candidates, it is natural to enforce them to be orthogonal to the residual vector $X\beta - y$, i.e., one should make use of them for reconstructing the overestimation $y$. This is the rationale behind the constraint: $\|X_{F_0}^T(X\beta - y)\|_\infty = 0$. The other advantage is when all correct features are chosen, the proposed algorithm can be shown to converge to the oracle solution. The detailed procedure is formally described in **Algorithm 1** below. Apparently, when $F_0^{(0)} = \varnothing$ and $N = 0$, the proposed method is identical to the standard Dantzig selector.

---

**Algorithm 1** Multi-Stage Dantzig Selector

---

**Require:** $F_0^{(0)}$, $\lambda$, $N$, $X$, $y$,
**Ensure:** $\hat{\beta}^{(N)}$, $F_0^{(N)}$
1: **while** i=0; i≤N; i++ **do**
2:     Obtain $\hat{\beta}^{(i)}$ by solving the problem (6) with $F_0 = F_0^{(i)}$
3:     Form $F_0^{(i+1)}$ as the index set of the $i + 1$ largest elements of $\hat{\beta}^{(i)}$.
4: **end while**

---

## 3    Main Results

### 3.1    Motivation

To motivate the proposed multi-stage algorithm, we first consider a simple case where some knowledge about the supporting features is known in advance. In standard Dantzig selector, we assume $F_0 = \varnothing$. If we assume that the features belonging to a set $F_0$ are known as supporting features, i.e., $F_0 \subset F$, we have the following result:

**Theorem 1.** *Assume that assumption 1 holds. Take $F_0 \subset F$ and $\lambda = \sigma\sqrt{2\log\left(\frac{m-s}{\eta_1}\right)}$ in the optimization problem* (6)*. If there exists some $l$ such that*

$$
\mu_{A,s+l}^{(p)} - \theta_{A,s+l,l}^{(p)}\left(\frac{|\bar{F}_0 - \bar{F}|}{l}\right)^{1-1/p} > 0
$$

*holds, then with a probability larger than $1 - \eta_1'$, the $l_p$-norm ($1 \le p \le \infty$) of the difference between $\hat{\beta}$, the solution of the problem* (6)*, and the oracle solution $\bar{\beta}$ is bounded as*

$$
\|\hat{\beta} - \bar{\beta}\|_p \le \frac{\left[1 + \left(\frac{|\bar{F}_0 - \bar{F}|}{l}\right)^{p-1}\right]^{1/p}(|\bar{F}_0 - \bar{F}| + l2^p)^{1/p}}{\mu_{A,s+l}^{(p)} - \theta_{A,s+l,l}^{(p)}\left(\frac{|\bar{F}_0 - \bar{F}|}{l}\right)^{1-1/p}}\sigma\sqrt{2\log\left(\frac{m-s}{\eta_1}\right)}
\tag{7}
$$

*and with a probability larger than $1 - \eta_1' - \eta_2'$, the $l_p$-norm ($1 \le p \le \infty$) of the difference between $\hat{\beta}$, the solution of the problem* (6) *and the true solution $\beta^*$ is bounded as*

$$
\begin{aligned}
\|\hat{\beta} - \beta^*\|_p \le \ & \frac{\left[1 + \left(\frac{|\bar{F}_0 - \bar{F}|}{l}\right)^{p-1}\right]^{1/p}(|\bar{F}_0 - \bar{F}| + l2^p)^{1/p}}{\mu_{A,s+l}^{(p)} - \theta_{A,s+l,l}^{(p)}\left(\frac{|\bar{F}_0 - \bar{F}|}{l}\right)^{1-1/p}}\sigma\sqrt{2\log\left(\frac{m-s}{\eta_1}\right)} + \\
& \frac{s^{1/p}}{\mu_{(X_F^T X_F)^{1/2},s}^{(p)}}\sigma\sqrt{2\log(s/\eta_2)}
\end{aligned}
\tag{8}
$$

It is clear that both bounds (for any $1 \le p \le \infty$) are monotonically increasing with respect to the value of $|\bar{F}_0 - \bar{F}|$. In other words, the larger $F_0$ is, the lower these bounds are. This coincides with our motivation that more knowledge about the supporting features can lead to a better signal estimation. Most related literatures directly estimate the bound of $\|\hat{\beta} - \beta^*\|_p$. Since $\beta^*$ may not be a feasible solution of problem (6), it is not easy to directly estimate the distance between $\hat{\beta}$ and $\beta^*$.

The bound in the inequality (8), which consists of two terms. Since $m \gg n \ge s$, we have $\sqrt{2\log((m-s)/\eta_1)} \gg \sqrt{2\log(s/\eta_2)}$ if $\eta_1 \approx \eta_2$. When $p = 2$, the following holds:

$$\mu_{A,s+l}^{(2)} - \theta_{A,s+l,l}^{(2)} \left( \frac{|\bar{F}_0 - \bar{F}|}{l} \right)^{1-1/2} \le \mu_{(X_F^T X_F)^{1/2},s}^{(2)}$$

since

$$\mu_{A,s+l}^{(2)} \le \mu_{A,s}^{(2)} \le \mu_{X_F^T X_F,s}^{(2)} \le \mu_{(X_F^T X_F)^{1/2},s}^{(2)}.$$

From the analysis in the next section, we can see that the first term is the upper bound of the distance from the optimizer to the oracle solution $\|\hat{\beta} - \bar{\beta}\|_p$ and the second term is the upper bound of the distance from the oracle solution to the true solution $\|\bar{\beta} - \beta^*\|_p$. Thus, the first term might be much larger than the second term.

## 3.2 Comparison with Dantzig Selector

We first compare our estimation bound with the one in [7] for $p = 2$. For convenience of comparison, we rewrite the theorem in [7] equivalently as:

**Theorem 2.** *Suppose $\beta \in \mathbb{R}^m$ is any s-sparse vector of parameters obeying $\delta_{2s} + \theta_{A,s,2s}^{(2)} < 1$. Setting $\lambda_p = \sigma\sqrt{2\log(m/\eta)}$ ($0 < \eta \le 1$), with a probability at least $1 - \eta(\pi\log m)^{-1/2}$, the solution of the standard Dantzig selector $\hat{\beta}_D$ obeys*

$$\|\hat{\beta}_D - \beta^*\|_2 \le \frac{4}{1 - \delta_{2s} - \theta_{A,s,2s}^{(2)}} s^{1/2} \sigma \sqrt{2\log(m/\eta)}, \tag{9}$$

*where $\delta_{2s} = \max(\rho_{A,2s}^{(2)} - 1, 1 - \mu_{A,2s}^{(2)})$.*

Theorem 1 also implies a bound estimation result for Dantzig selector by letting $F_0 = \varnothing$ and $p = 2$. Specifically, we set $F_0 = \varnothing$, $N = 0$, and $\lambda = \sigma\sqrt{2\log\left(\frac{m-s}{\eta_1}\right)}$ in the multi-stage method, and set $p = 2$, $l = s$, $\eta_1 = \frac{m-s}{m}\eta$, and $\eta_2 = \frac{s}{m}\eta$ for a convenient of comparison with Theorem 1. If follows that with probability larger than $1 - \eta(\pi\log m)^{-1/2}$, the following bound holds:

$$\|\hat{\beta} - \beta^*\|_2 \le \left( \frac{\sqrt{10}}{\mu_{A,2s}^{(2)} - \theta_{A,2s,s}^{(2)}} + \frac{1}{\mu_{(X_F^T X_F)^{1/2},s}^{(2)}} \right) s^{1/2} \sigma \sqrt{2\log(m/\eta)}. \tag{10}$$

It is easy to verify that

$$1 - \delta_{2s} - \theta_{A,s,2s}^{(2)} \le \mu_{A,2s}^{(2)} - \theta_{A,2s,s}^{(2)} \le \mu_{A,2s}^{(2)} \le \mu_{(X_F^T X_F),s}^{(2)} = \left( \mu_{(X_F^T X_F)^{1/2},s}^{(2)} \right)^2 \le \mu_{(X_F^T X_F)^{1/2},s}^{(2)} \le 1.$$

Thus, the bound in (10) is comparable to the one in (9). In the following, we compare the performance bound of the proposed multi-stage method ($N > 0$) with the one in (10).

## 3.3 Feature Selection

The estimation bounds in Theorem 1 assume that a set $F_0$ is given. In this section, we show how the supporting set can be estimated. Similar to previous work [5, 19], $|\beta_j^*|$ for $j \in F$ is required to be larger than a threshold value. As is clear from the proof in the next section, the threshold value mainly depends on the value of $\|\hat{\beta} - \beta^*\|_\infty$. We essentially employ the result with $p = \infty$ in Theorem 1 to estimate the threshold value. In the following, we first consider the simple case when $N = 0$. We have shown in the last section that the estimation bound in this case is similar to the one for Dantzig selector.

**Theorem 3.** *Under the assumption 1, if there exists an index set $J$ such that $|\beta_j^*| > \alpha_0$ for any $j \in J$ and there exists a nonempty set*

$$\Omega = \{l \mid \mu_{A,s+l}^{(\infty)} - \theta_{A,s+l,l}^{(\infty)} \left(\frac{s}{l}\right) > 0\}$$

*where*

$$\alpha_0 = 4 \min_{l \in \Omega} \frac{\max\left(1, \frac{s}{l}\right)}{\mu_{A,s+l}^{(\infty)} - \theta_{A,s+l,l}^{(\infty)} \left(\frac{s}{l}\right)} \sigma \sqrt{2 \log\left(\frac{m-s}{\eta_1}\right)} + \frac{1}{\mu_{(X_F^T X_F)^{1/2},s}^{(\infty)}} \sigma \sqrt{2 \log(s/\eta_2)},$$

*then taking $F_0 = \varnothing$, $N = 0$, $\lambda = \sigma \sqrt{2 \log\left(\frac{m-s}{\eta_1}\right)}$ into the problem (6) (equivalent to Dantzig selector), the largest $|J|$ elements of $\hat{\beta}_{std}$ (or $\hat{\beta}^{(0)}$) belong to $F$ with probability larger than $1 - \eta_1' - \eta_2'$.*

The theorem above indicates that under the given condition, if $\min_{j \in J} |\beta_j^*| > \mathcal{O}(\sigma \sqrt{\log m})$ (assuming that there exists $l \geq s$ such that $\mu_{A,s+l}^{(\infty)} - \theta_{A,s+l,l}^{(\infty)} \left(\frac{s}{l}\right) > 0$), then with high probability the selected $|J|$ features by Dantzig selector belong to the true supporting set. In particular, if $|J| = s$, then the consistency of feature selection is achieved. The result above is comparable to the ones for other feature selection algorithms, including LASSO [5, 22], greedy least squares regression [16, 8, 19], two stage LASSO [20], and adaptive forward-backward greedy algorithm [18]. In all these algorithms, the condition $\min_{j \in F} |\beta_j^*| \geq C\sigma \sqrt{\log m}$ is required, since the noise level is $\mathcal{O}(\sigma \sqrt{\log m})$ [18]. Because $C$ is always a coefficient in terms of the covariance matrix $XX^T$ (or the feature matrix $X$), it is typically treated as a constant term; see the literature listed above.

Next, we show that the condition $|\beta_j^*| > \alpha_0$ in Theorem 3 can be relaxed by the proposed multi-stage procedure with $N > 0$, as summarized in the following theorem:

**Theorem 4.** *Under the assumption 1, if there exists a nonempty set*

$$\Omega = \{l \mid \mu_{A,s+l}^{(\infty)} - \theta_{A,s+l,l}^{(\infty)} \left(\frac{s}{l}\right) > 0\}$$

*and there exists a set $J$ such that $|supp_{\alpha_i}(\beta_J^*)| > i$ holds for all $i \in \{0, 1, ..., |J| - 1\}$, where*

$$\alpha_i = 4 \min_{l \in \Omega} \frac{\max\left(1, \frac{s-i}{l}\right)}{\left\{\mu_{A,s+l}^{(\infty)} - \theta_{A,s+l,l}^{(\infty)} \left(\frac{s-i}{l}\right)\right\}} \sigma \sqrt{2 \log\left(\frac{m-s}{\eta_1}\right)} + \frac{1}{\mu_{(X_F^T X_F)^{1/2},s}^{(\infty)}} \sigma \sqrt{2 \log(s/\eta_2)},$$

*then taking $F_0^{(0)} = \varnothing$, $\lambda = \sigma \sqrt{2 \log\left(\frac{m-s}{\eta_1}\right)}$ and $N = |J| - 1$ into **Algorithm** 1, the solution after $N$ iterations satisfies $F_0^{(N)} \subset F$ (i.e. $|J|$ correct features are selected) with probability larger than $1 - \eta_1' - \eta_2'$.*

Assume that one aims to select $N$ correct features by the standard Dantzig selector and the multi-stage method. These two theorems show that the standard Dantzig selector requires that at least $N$ of $|\beta_j^*|$'s with $j \in F$ are larger than the threshold value $\alpha_0$, while the proposed multi-stage method requires that at least $i$ of the $|\beta_j^*|$'s are larger than the threshold value $\alpha_{i-1}$, for $i = 1, \cdots, N$. Since $\{\alpha_j\}$ is a strictly decreasing sequence satisfying for some $l \in \Omega$,

$$\alpha_{i-1} - \alpha_i > \frac{4\theta_{A,s+l,l}^{(\infty)}}{l \left(\mu_{A,s+l}^{(\infty)} - \theta_{A,s+l,l}^{(\infty)} \left(\frac{s-i}{l}\right)\right)^2} \sigma \sqrt{2 \log\left(\frac{m-s}{\eta_1}\right)},$$

the proposed multi-stage method requires a strictly weaker condition for selecting $N$ correct features than the standard Dantzig selector.

### 3.4 Signal Recovery

In this section, we derive the estimation bound of the proposed multi-stage method by combing results from Theorems 1, 3, and 4.

**Theorem 5.** *Under the assumption 1, if there exists $l$ such that*

$$\mu_{A,s+l}^{(\infty)} - \theta_{A,s+l,l}^{(\infty)}\left(\frac{s}{l}\right) > 0 \quad and \quad \mu_{A,2s}^{(p)} - \theta_{A,2s,s}^{(p)} > 0,$$

*and there exists a set $J$ such that $|supp_{\alpha_i}(\beta_J^*)| > i$ holds for all $i \in \{0, 1, ..., |J| - 1\}$, where the $\alpha_i$'s are defined in Theorem 4, then*

*(1) taking $F_0 = \varnothing$, $N = 0$ and $\lambda = \sigma\sqrt{2\log\left(\frac{m-s}{\eta_1}\right)}$ into **Algorithm** 1, with probability larger than $1 - \eta_1' - \eta_2'$, the solution of the Dantzig selector $\hat{\beta}_D$ (i.e, $\hat{\beta}^{(0)}$) obeys:*

$$\|\hat{\beta}_D - \beta^*\|_p \le \frac{(2^{p+1}+2)^{1/p}s^{1/p}}{\mu_{A,2s}^{(p)} - \theta_{A,2s,s}^{(p)}}\sigma\sqrt{2\log\left(\frac{m-s}{\eta_1}\right)} + \frac{s^{1/p}}{\mu_{(X_F^T X_F)^{1/2},s}^{(p)}}\sigma\sqrt{2\log(s/\eta_2)}; \quad (11)$$

*(2) taking $F_0 = \varnothing$, $N = |J|$ and $\lambda = \sigma\sqrt{2\log\left(\frac{m-s}{\eta_1}\right)}$ into **Algorithm** 1, with probability larger than $1 - \eta_1' - \eta_2'$, the solution of the multi-stage method $\hat{\beta}_{mul}$ (i.e., $\hat{\beta}^{(N)}$) obeys:*

$$\|\hat{\beta}_{mul} - \beta^*\|_p \le \frac{(2^{p+1}+2)^{1/p}(s-N)^{1/p}}{\mu_{A,2s-N}^{(p)} - \theta_{A,2s-N,s-N}^{(p)}}\sigma\sqrt{2\log\left(\frac{m-s}{\eta_1}\right)} + \frac{s^{1/p}}{\mu_{(X_F^T X_F)^{1/2},s}^p}\sigma\sqrt{2\log(s/\eta_2)}.$$
$$(12)$$

Similar to the analysis in Theorem 1, the first term (i.e., the distance from $\hat{\beta}$ to the oracle solution $\bar{\beta}$) dominates in the estimated bounds. Thus, the performance of the multi-stage method approximately improved the standard Dantzig selector from $Cs^{1/p}\sqrt{\log m}\sigma$ to $C(s-N)^{1/p}\sqrt{\log m}\sigma$. When $p = 2$, our estimation has the same order as the greedy least squares regression algorithm [19] and the adaptive forward-backward greedy algorithm [18].

### 3.5 The Oracle Solution

The oracle solution is the minimum-variance unbiased estimator of the true solution given the noisy observation. We show in the following theorem that the proposed method can obtain the oracle solution with high probability under certain conditions:

**Theorem 6.** *Under the assumption 1, if there exists $l$ such that $\mu_{A,s+l}^{(\infty)} - \theta_{A,s+l,l}^{(\infty)}\left(\frac{s-i}{l}\right) > 0$, and the supporting set $F$ of $\beta^*$ satisfies $|supp_{\alpha_i}(\beta_F^*)| > i$ for all $i \in \{0, 1, ..., s - 1\}$, where the $\alpha_i$'s are defined in Theorem 4, then taking $F_0 = \varnothing$, $N = s$ and $\lambda = \sigma\sqrt{2\log\left(\frac{m-s}{\eta_1}\right)}$ into **Algorithm** 1, the oracle solution can be achieved, i.e. $F_0^{(N)} = F$ and $\hat{\beta}^{(N)} = \bar{\beta}$ with probability larger than $1 - \eta_1' - \eta_2'$.*

The theorem above shows that when the nonzero elements of the true coefficients vector $\beta^*$ are large enough, the oracle solution can be achieved with high probability.

## 4 Simulation Study

We have performed simulation studies to verify our theoretical analysis. Our comparison includes two aspects: signal recovery accuracy and feature selection accuracy. The signal recovery accuracy is measured by the relative signal error: $SRA = \|\hat{\beta} - \beta^*\|_2/\|\beta^*\|_2$, where $\hat{\beta}$ is the solution of a specific algorithm. The feature selection accuracy is measured by the percentage of correct features selected: $FSA = |\hat{F} \cap F|/|F|$, where $\hat{F}$ is the estimated feature candidate set.

We generate an $n \times m$ random matrix $X$. Each element of $X$ follows an independent standard Gaussian distribution $N(0, 1)$. We then normalize the length of the columns of $X$ to be 1. The $s-$sparse original signal $\beta^*$ is generated with $s$ nonzero elements independently uniformly distributed from

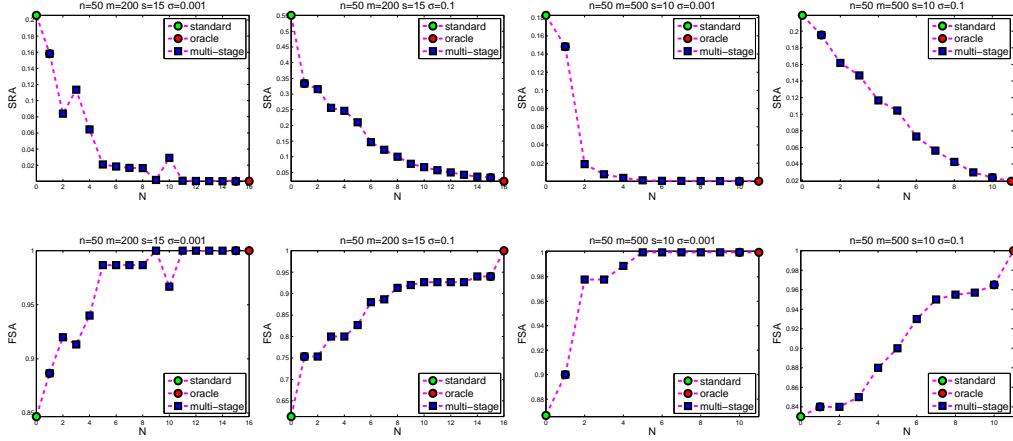

Figure 1: Numerical simulation. We compare the solutions of the standard Dantzig selector method ($N = 0$), the proposed method for different values of $N$, and the oracle solution. The $SRA$ and $FSA$ comparisons are reported on the top row and the bottom row, respectively. The starting point of each curve records the $SRA$ (or $FSA$) value of the standard Dantzig selector method; the ending point records the value of the oracle solution; the middle part of each curve records the results by the proposed method for different values of $N$.

$[-10, 10]$. We for $y$ by $y = X\beta^* + \epsilon$, where the noise vector $\epsilon$ is generated by the Gaussian distribution $N(0, \sigma^2 I)$. For a fair comparison, we choose the same $\lambda = \sigma\sqrt{2\log m}$ in both algorithms. The following experiments are repeated 20 times and we report their average performance.

We run the proposed algorithm with $F_0^{(0)} = \varnothing$ and output the $\hat{\beta}^{(N)}$'s. Note that the solution of the standard Dantzig selector algorithm is equivalent to $\hat{\beta}^{(0)}$ with $N = 0$. We report the $SRA$ curve of $\hat{\beta}^{(N)}$ with respect to $N$ in the top row of Figure 1. Based on $\hat{\beta}^{(N)}$, we compute the supporting set $\hat{F}^{(N)}$ as the index of the $N$ largest entries in $\hat{\beta}^{(N)}$. Note that the supporting set we compute here is different from the supporting set $\hat{F}_0^{(N)}$ which only contains the $N$ largest feature indexes. The bottom row of Figure 1 shows the $FSA$ curve with respect to $N$. We can observe from Figure 1 that 1) the multi-stage method obtains a solution with a smaller distance to the original signal than the standard Dantzig selector method; 2) the multi-stage method selects a larger percentage of correct features than the standard Dantzig selector method; 3) the multi-stage method can achieve the oracle solution. Overall, the recovery accuracy curve increases with an increasing value of $N$ and the feature selection accuracy curve is decreasing with an increasing value of $N$.

# 5  Conclusion

In this paper, we propose a multi-stage Dantzig selector method which iteratively selects the supporting features and recovers the original signal. The proposed method makes use of the information of supporting features to estimate the signal and simultaneously makes use of the information of the estimated signal to select the supporting features. Our theoretical analysis shows that the proposed method improves upon the standard Dantzig selector in both signal recovery and supporting feature selection. The final numerical simulation validates our theoretical analysis.

Since the multi-stage procedure can improve the Dantzig selector, one natural question is whether the analysis can be extended to other related techniques such as LASSO. The two-stage LASSO has been shown to outperform the standard LASSO. We plan to extend our analysis for multi-stage LASSO in the future. In addition, we plan to improve the proposed algorithm by adopting stopping rules similar to the ones recently proposed in [3, 19, 21].

**Acknowledgments**

This work was supported by NSF IIS-0612069, IIS-0812551, CCF-0811790, IIS-0953662, and NGA HM1582-08-1-0016.

# References

[1] P. J. Bickel, Y. Ritov, and A. B. Tsybakov. Simultaneous analysis of Lasso and Dantzig selector. *Annals of Statistics*, 37:1705–1732, 2009.

[2] F. Bunea, A. Tsybakov, and M. Wegkamp. Sparsity oracle inequalities for the Lasso. *Electronic Journal of Statistics*, 2007.

[3] T. Cai and L. Wang. Orthogonal matching pursuit for sparse signal reconvery. *Technical Report*, 2010.

[4] T. Cai, G. Xu, and J. Zhang. On recovery of sparse signals via $l_1$ minimization. *IEEE Transactions on Information Theory*, 55(7):3388–3397, 2009.

[5] E. J. Candes and Y. Plan. Near-ideal model selection by $l_1$ minimization. *Annals of Statistics*, 37:2145–2177, 2006.

[6] E. J. Candes and T. Tao. Decoding by linear programming. *IEEE Transactions on Information Theory*, 51(12):4203–4215, 2005.

[7] E. J. Candes and T. Tao. The Dantzig selector: Statistical estimation when $p$ is much larger than $n$. *Annals of Statistics*, 35:2313, 2007.

[8] D. L. Donoho, M. Elad, and V. N. Temlyakov. Stable recovery of sparse overcomplete representations in the presence of noise. *IEEE Transactions on Information Theory*, pages 6–18, 2006.

[9] G. M. James, P. Radchenko, and J. Lv. DASSO: connections between the Dantzig selector and Lasso. *Journal of The Royal Statistical Society Series B*, 71(1):127–142, 2009.

[10] V. Koltchinskii and M. Yuan. Sparse recovery in large ensembles of kernel machines on-line learning and bandits. *COLT*, pages 229–238, 2008.

[11] K. Lounici. Sup-norm convergence rate and sign concentration property of Lasso and Dantzig esti mators. *Electronic Journal of Statistics*, 2:90–102, 2008.

[12] N. Meinshausen, P. Bhlmann, and E. Zrich. High dimensional graphs and variable selection with the Lasso. *Annals of Statistics*, 34:1436–1462, 2006.

[13] P. Ravikumar, G. Raskutti, M. J. Wainwright, and B. Yu. Model selection in gaussian graphical models: High-dimensional consistency of $l_1$-regularized MLE. pages 1329–1336, 2008.

[14] J. Romberg. The Dantzig selector and generalized thresholding. *CISS*, pages 22–25, 2008.

[15] R. Tibshirani. Regression shrinkage and selection via the Lasso. *Journal of the Royal Statistical Society: Series B*, 58(1):267–288, 1996.

[16] J. A. Tropp. Greed is good: Algorithmic results for sparse approximation. *IEEE Transactions on Information Theory*, 50:2231–2242, 2004.

[17] M. J. Wainwright. Sharp thresholds for noisy and high-dimensional recovery of sparsity using $l_1$-constrained quadratic programming (Lasso). *IEEE Transactions on Information Theory*, pages 2183–2202, 2009.

[18] T. Zhang. Adaptive forward-backward greedy algorithm for sparse learning with linear models. *NIPS*, pages 1921–1928, 2008.

[19] T. Zhang. On the consistency of feature selection using greedy least squares regression. *Journal of Machine Learning Reserch*, 10:555–568, 2009.

[20] T. Zhang. Some sharp performance bounds for least squares regression with $l_1$ regularization. *Annals of Statistics*, 37:2109, 2009.

[21] T. Zhang. Sparse recovery with orthogonal matching pursuit under RIP. *arXiv:1005.2249*, 2010.

[22] P. Zhao and B. Yu. On model selection consistency of Lasso. *Journal of Machine Learning Reserch*, 7:2541–2563, 2006.

[23] S. Zhou. Thresholding procedures for high dimensional variable selection and statistical estimation. *NIPS*, pages 2304–2312, 2009.

[24] H. Zou. The adaptive Lasso and its oracle properties. *Journal of the American Statistical Association*, 101:1418–1429, 2006.

